# Boosting and Maximum Likelihood for Exponential Models

**Guy Lebanon**
School of Computer Science
Carnegie Mellon University
Pittsburgh, PA 15213
lebanon@cs.cmu.edu

**John Lafferty**
School of Computer Science
Carnegie Mellon University
Pittsburgh, PA 15213
lafferty@cs.cmu.edu

## Abstract

We derive an equivalence between AdaBoost and the dual of a convex optimization problem, showing that the only difference between minimizing the exponential loss used by AdaBoost and maximum likelihood for exponential models is that the latter requires the model to be normalized to form a conditional probability distribution over labels. In addition to establishing a simple and easily understood connection between the two methods, this framework enables us to derive new regularization procedures for boosting that directly correspond to penalized maximum likelihood. Experiments on UCI datasets support our theoretical analysis and give additional insight into the relationship between boosting and logistic regression.

## 1  Introduction

Several recent papers in statistics and machine learning have been devoted to the relationship between boosting and more standard statistical procedures such as logistic regression. In spite of this activity, an easy-to-understand and clean connection between these different techniques has not emerged. Friedman, Hastie and Tibshirani [7] note the similarity between boosting and stepwise logistic regression procedures, and suggest a least-squares alternative, but view the loss functions of the two problems as different, leaving the precise relationship between boosting and maximum likelihood unresolved. Kivinen and Warmuth [8] note that boosting is a form of "entropy projection," and Lafferty [9] suggests the use of Bregman distances to approximate the exponential loss. Mason *et al.* [10] consider boosting algorithms as functional gradient descent and Duffy and Helmbold [5] study various loss functions with respect to the PAC boosting property. More recently, Collins, Schapire and Singer [2] show how different Bregman distances precisely account for boosting and logistic regression, and use this framework to give the first convergence proof of AdaBoost. However, in this work the two methods are viewed as minimizing different loss functions. Moreover, the optimization problems are formulated in terms of a reference distribution consisting of the zero vector, rather than the empirical distribution of the data, making the interpretation of this use of Bregman distances problematic from a statistical point of view.

In this paper we present a very basic connection between boosting and maximum likelihood for exponential models through a simple convex optimization problem. In this setting, it is

seen that the only difference between AdaBoost and maximum likelihood for exponential models, in particular logistic regression, is that the latter requires the model to be normalized to form a probability distribution. The two methods minimize the same extended Kullback-Leibler divergence objective function subject to the same feature constraints. Using information geometry, we show that projecting the exponential loss model onto the simplex of conditional probability distributions gives precisely the maximum likelihood exponential model with the specified sufficient statistics. In many cases of practical interest, the resulting models will be identical; in particular, as the number of features increases to fit the training data the two methods will give the same classifiers. We note that throughout the paper we view boosting as a procedure for minimizing the exponential loss, using either parallel or sequential update algorithms as in [2], rather than as a forward stepwise procedure as presented in [7] or [6].

Given the recent interest in these techniques, it is striking that this connection has gone unobserved until now. However in general, there may be many ways of writing the constraints for a convex optimization problem, and many different settings of the Lagrange multipliers (or Kuhn-Tucker vectors) that represent identical solutions. The key to the connection we present here lies in the use of a particular non-standard presentation of the constraints. When viewed in this way, there is no need for special-purpose Bregman distances to give a unified account of boosting and maximum likelihood, as we only make use of the standard Kullback-Leibler divergence. But our analysis gives more than a formal framework for understanding old algorithms; it also leads to new algorithms for regularizing AdaBoost, which is required when the training data is noisy. In particular, we derive a regularization procedure for AdaBoost that directly corresponds to penalized maximum likelihood using a Gaussian prior. Experiments on UCI data support our theoretical analysis, demonstrate the effectiveness of the new regularization method, and give further insight into the relationship between boosting and maximum likelihood exponential models.

## 2   Notation

Let $\mathcal{X}$ and $\mathcal{Y}$ be finite sets. We denote by $\mathcal{M} = \{m : \mathcal{X} \times \mathcal{Y} \to \mathbb{R}_+\}$ the set of non-negative measures on $\mathcal{X} \times \mathcal{Y}$, and by $\Delta \subset \mathcal{M}$ the set of conditional probability distributions, $\Delta = \{m \in \mathcal{M} : \sum_{y \in \mathcal{Y}} m(x, y) = 1, \text{ for each } x \in \mathcal{X}\}$. For $m \in \mathcal{M}$, we will overload the notation $m(x, y)$ and $m(y \,|\, x)$; the latter will be suggestive of a conditional probability distribution, but in general it need not be normalized. Let $f_j : \mathcal{X} \times \mathcal{Y} \to \mathbb{R}$, $j = 1, \dots, m$, be given functions, which we will refer to as *features*. These will correspond to the *weak learners* in boosting, and to the *sufficient statistics* in an exponential model. Suppose that we have data $\{(x_i, y_i)\}_{i=1}^n$ with empirical distribution $\widetilde{p}(x, y)$ and marginal $\widetilde{p}(x)$; thus, $\widetilde{p}(x, y) = \frac{1}{n} \sum_{i=1}^n \delta(x_i, x)\,\delta(y_i, y)$. We assume, without loss of generality, that $\widetilde{p}(x) > 0$ for all $x$. Throughout the paper, we assume (for notational convenience) that the training data has the following property.

*Consistent Data Assumption*. For each $x \in \mathcal{X}$ with $\widetilde{p}(x) > 0$, there is a unique $y \in \mathcal{Y}$ for which $\widetilde{p}(y \,|\, x) > 0$. This $y$ will be denoted $\widetilde{y}(x)$.

For most data sets of interest, each $x$ appears only once, so that the assumption trivially holds. However, if $x$ appears more than once, we require that it is labeled consistently. We make this assumption mainly to correspond with the conventions used to present boosting algorithms; it is not essential to what follows.

Given $f_j$, we define the exponential model $q_\lambda(y \,|\, x)$, for $\lambda \in \mathbb{R}^m$, by $q_\lambda(y \,|\, x) = \frac{1}{\sum_y e^{\langle \lambda, f(x,y) \rangle}} e^{\langle \lambda, f(x,y) \rangle}$ where $\langle \lambda, f(x,y) \rangle = \sum_{j=1}^m \lambda_j f_j(x, y)$. The maximum likelihood estimation problem is to determine parameters $\lambda$ that maximize the conditional log-

likelihood $\ell(\lambda) = \sum_{x,y} \widetilde{p}(x,y) \log q_\lambda(y \mid x)$ or minimize the log loss $-\ell(\lambda)$. The objective function to be minimized in the multi-label boosting algorithm AdaBoost.M2 [2] is the exponential loss given by $\mathcal{E}_{\mathrm{M2}}(\lambda) = \sum_{i=1}^{n} \sum_{y \neq y_i} e^{\langle \lambda, f(x_i,y) - f(x_i,y_i) \rangle}$. As has been often noted, the log loss and the exponential loss are qualitatively different. The exponential loss grows exponentially with increasing negative "margin," while the log loss grows linearly.

## 3  Correspondence Between AdaBoost and Maximum Likelihood

Since we are working with unnormalized models we make use of the extended conditional Kullback-Leibler divergence or $I$-divergence, given by

$$D(p,q) \overset{\text{def}}{=} \sum_x \widetilde{p}(x) \sum_y \left( p(y \mid x) \log \frac{p(y \mid x)}{q(y \mid x)} - p(y \mid x) + q(y \mid x) \right)$$

defined on $\mathcal{M} \times \mathcal{M}$ (possibly taking on the value $\infty$). Note that if $p(\cdot \mid x) \in \Delta$ and $q(\cdot \mid x) \in \Delta$ then this becomes the more familiar KL divergence for probabilities. Let features $f_j$ and a fixed default distribution $q_0 \in \mathcal{M}$ be given. We define $\mathcal{F}(\widetilde{p}, f) \subset \mathcal{M}$ as

$$\mathcal{F}(\widetilde{p}, f) = \{ p \in \mathcal{M} : \sum_x \widetilde{p}(x) \sum_y p(y \mid x) \, (f_j(x,y) - E_{\widetilde{p}}[f_j \mid x]) = 0, \text{ all } j \}. \quad (1)$$

Since $\widetilde{p} \in \mathcal{F}$, this set is non-empty. Note that under the consistent data assumption, we have that $E_{\widetilde{p}}[f \mid x] = f(x, \widetilde{y}(x))$. Consider now the following two convex optimization problems, labeled $P_1$ and $P_2$.

$(P_1)$    *minimize*    $D(p, q_0)$
        *subject to*    $p \in \mathcal{F}(\widetilde{p}, f)$

$(P_2)$    *minimize*    $D(p, q_0)$
        *subject to*    $p \in \mathcal{F}(\widetilde{p}, f)$
                   $p \in \Delta$

Thus, problem $P_2$ differs from $P_1$ only in that the solution is required to be normalized. As we'll show, the dual problem $P_1^*$ corresponds to AdaBoost, and the dual problem $P_2^*$ corresponds to maximum likelihood for exponential models.

This presentation of the constraints is the key to making the correspondence between AdaBoost and maximum likelihood. Note that the constraint $\sum_x \widetilde{p}(x) \sum_y p(y \mid x) f(x,y) = E_{\widetilde{p}}[f]$, which is the usual presentation of the constraints for maximum likelihood (as dual to maximum entropy), doesn't make sense for unnormalized models, since the two sides of the equation may not be "on the same scale." Note further that attempting to rescale by dividing by the mass of $p$ to get

$$\sum_x \widetilde{p}(x) \frac{\sum_y p(y \mid x) \, f(x,y)}{\sum_y p(y \mid x)} = E_{\widetilde{p}}[f]$$

would yield *nonlinear* constraints.

We now derive the dual problems formally; the following section gives a precise statement of the duality result. To derive the dual problem $P_1^*$, we calculate the Lagrangian as

$$\mathcal{L}_1(p, \lambda) = \sum_x \widetilde{p}(x) \sum_y p(y \mid x) \left( \log \frac{p(y \mid x)}{q_0(y \mid x)} - 1 - \langle \lambda, f(x,y) - E_{\widetilde{p}}[f \mid x] \rangle \right).$$

For $\lambda \in \mathbb{R}^m$, the connecting equation $q_\lambda \overset{\text{def}}{=} \arg\min_{p \in \mathcal{M}} \mathcal{L}_1(p, \lambda)$ is given by $q_\lambda(y \mid x) = q_0(y \mid x) e^{\langle \lambda, f(x,y) - E_{\widetilde{p}}[f \mid x] \rangle}$. Thus, the dual function $\widehat{h}_1(\lambda) = \mathcal{L}_1(q_\lambda, \lambda)$ is given by

$$\widehat{h}_1(\lambda) = -\sum_x \widetilde{p}(x) \sum_y q_0(y \mid x) e^{\langle \lambda, f(x,y) - E_{\widetilde{p}}[f \mid x] \rangle}. \quad (2)$$

The dual problem is to determine $\lambda^\star = \arg\max_\lambda \widehat{h}(\lambda)$. To derive the dual for $P_2$, we simply add additional Lagrange multipliers $\mu_x$ for the constraints $\sum_y p(y \mid x) = 1$.

### 3.1 Special cases

It is now straightforward to derive various boosting and logistic regression problems as special cases of the above optimization problems.

*Case 1: AdaBoost.M2.* Take $q_0(y \mid x) = 1$. Then the dual problem $\max_\lambda \widehat{h}_1(\lambda)$ is equivalent to computing $\lambda^\star = \arg\min_\lambda \sum_i \sum_{y \neq y_i} e^{\langle \lambda, f(x_i, y) - f(x_i, y_i) \rangle}$ which is the optimization problem of AdaBoost.M2.

*Case 2: Binary AdaBoost.* In addition to the assumptions for the previous case, now assume that $y \in \{-1, +1\}$, and take $f_j(x, y) = \frac{1}{2} y f_j(x)$. Then the dual problem is given by $\lambda^\star = \arg\min_\lambda \sum_i e^{-y_i \langle \lambda, f(x_i) \rangle}$ which is the optimization problem of binary AdaBoost.

*Case 3: Maximum Likelihood for Exponential Models.* In this case we take the same setup as for AdaBoost.M2 but add the additional normalization constraints: $\sum_y p(y \mid x_i) = 1$, $i = 1, \ldots, n$. If these constraints are satisfied, then the other constraints take the form $\sum_x \widetilde{p}(x) \sum_y p(y \mid x) f_j(x, y) = \sum_{x,y} \widetilde{p}(x, y) f_j(x, y)$ and the connecting equation becomes $q_\lambda(y \mid x) = \frac{1}{Z_x} q_0(y \mid x) e^{\langle \lambda, f(x,y) \rangle}$ were $Z_x$ is the normalizing term $Z_x = \sum_y q_0(y|x) e^{\langle \lambda, f(x,y) \rangle}$, which corresponds to setting the Lagrange multiplier $\mu_x$ to the appropriate value. In this case, after a simple calculation the dual problem is seen to be $\widehat{h}_2(\lambda) = \sum_x \widetilde{p}(x, y) \log q_\lambda(y \mid x)$ which corresponds to maximum likelihood for a conditional exponential model with sufficient statistics $f_j(x, y)$.

*Case 4: Logistic Regression.* Returning to the case of binary AdaBoost, we see that when we add normalization constraints as above, the model is equivalent to binary logistic regression, since $q_\lambda(1 \mid x) = \frac{1}{1 + e^{-\langle \lambda, f(x) \rangle}}$. We note that it is not necessary to scale the features by a constant factor here, as in [7]; the correspondence between logistic regression and boosting is direct.

### 3.2 Duality

Let $\mathcal{Q}_1$ and $\mathcal{Q}_2$ be defined as the following exponential families:

$$\mathcal{Q}_1(q_0, f) = \{q \in \mathcal{M} : q(y \mid x) = q_0(y \mid x) e^{\langle \lambda, f(x,y) - f(x, \tilde{y}(x)) \rangle}, \lambda \in \mathbb{R}^m\}$$
$$\mathcal{Q}_2(q_0, f) = \{q \in \Delta : q(y \mid x) \propto q_0(y \mid x) e^{\langle \lambda, f(x,y) \rangle}, \lambda \in \mathbb{R}^m\}.$$

Thus $\mathcal{Q}_1$ is unnormalized while $\mathcal{Q}_2$ is normalized. We now define the boosting solution $q^\star_{boost}$ and maximum likelihood solution $q^\star_{ml}$ as

$$q^\star_{boost} = \arg\min_{q \in \overline{\mathcal{Q}_1}} \sum_x \widetilde{p}(x) \sum_y q(y \mid x) \quad q^\star_{ml} = \arg\max_{q \in \overline{\mathcal{Q}_2}} \sum_x \widetilde{p}(x) \log q(\tilde{y} \mid x)$$

where $\overline{Q}$ denotes the closure of the set $Q \subset \mathcal{M}$. The following theorem corresponds to Proposition 4 of [3] for the usual KL divergence; in [4] the duality theorem is proved for a general class of Bregman distances, including the extended KL divergence as a special case. Note that we do not work with divergences such as $D(\vec{0}, q)$ as in [2], but rather $D(\widetilde{p}, q)$, which is more natural and interpretable from a statistical point-of-view.

**Theorem.** *Suppose that $D(\widetilde{p}, q_0) < \infty$. Then $q^\star_{boost}$ and $q^\star_{ml}$ exist, are unique, and satisfy*

$$
\begin{aligned}
q^\star_{boost} &= \arg\min_{p \in \mathcal{F}} D(p, q_0) = \arg\min_{q \in \overline{\mathcal{Q}_1}} D(\widetilde{p}, q) \\
q^\star_{ml} &= \arg\min_{p \in \mathcal{F} \cap \Delta} D(p, q_0) = \arg\min_{q \in \overline{\mathcal{Q}_2}} D(\widetilde{p}, q).
\end{aligned}
$$

*Moreover, $q^\star_{ml}$ is computed in terms of $q^\star_{boost}$ as $q^\star_{ml} = \arg\min_{p \in \mathcal{F} \cap \Delta} D(p, q^\star_{boost})$.*

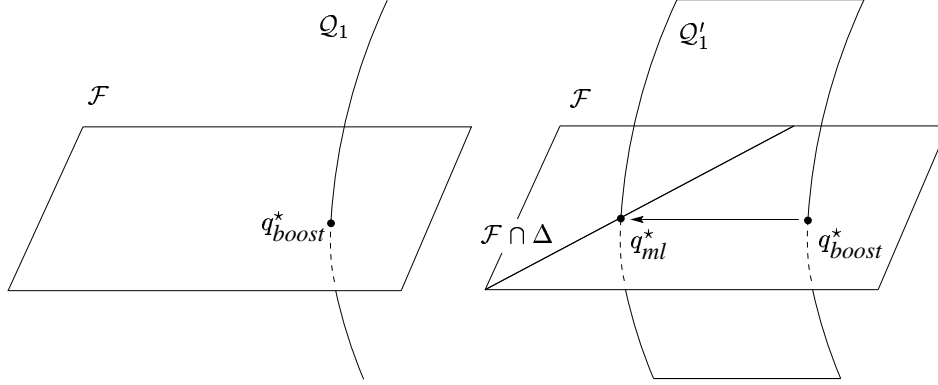

Figure 1: Geometric view of the duality theorem. Minimizing the exponential loss finds the member of $\mathcal{Q}_1$ that intersects with the feasible set of measures satisfying the moment constraints (left). When we impose the additional constraint that each conditional distribution $q_\lambda(y \mid x)$ must be normalized, we introduce a Lagrange multiplier for each training example $x$, giving a higher-dimensional family $\mathcal{Q}_1'$. By the duality theorem, projecting the exponential loss solution onto the intersection of the feasible set with the simplex gives the maximum likelihood solution.

This result has a simple geometric interpretation. The unnormalized exponential family $\mathcal{Q}_1$ intersects the feasible set of measures $\mathcal{F}$ satisfying the constraints (1) at a single point. The algorithms presented in [2] determine this point, which is the exponential loss solution $q_{boost}^\star = \arg\min_{q \in \overline{\mathcal{Q}_1}} D(\widetilde{p}, q)$ (see Figure 1, left).

On the other hand, maximum conditional likelihood estimation for an exponential model with the same features is equivalent to the problem $q_{ml}^\star = \arg\min_{q \in \overline{\mathcal{Q}_1'}} D(\widetilde{p}, q)$ where $\mathcal{Q}_1'$ is the exponential family with additional Lagrange multipliers, one for each normalization constraint. The feasible set for this problem is $\mathcal{F} \cap \Delta$. Since $\mathcal{F} \cap \Delta \subset \mathcal{F}$, by the Pythagorean equality we have that $q_{ml}^\star = \arg\min_{p \in \mathcal{F} \cap \Delta} D(p, q_{boost}^\star)$ (see Figure 1, right).

## 4 Regularization

Minimizing the exponential loss or the log loss on real data often fails to produce finite parameters. Specifically, this happens when for some feature $f_j$

$$f_j(x, y) - f_j(x, \widetilde{y}(x)) \geq 0 \text{ for all } y \text{ and } x \text{ with } \widetilde{p}(x) > 0 \qquad (3)$$
$$\text{or} \quad f_j(x, y) - f_j(x, \widetilde{y}(x)) \leq 0 \text{ for all } y \text{ and } x \text{ with } \widetilde{p}(x) > 0.$$

This is especially harmful since often the features for which (3) holds are the most important for the purpose of discrimination. Of course, even when (3) does not hold, models trained by maximum likelihood or the exponential loss can overfit the training data. A standard regularization technique in the case of maximum likelihood employs parameter priors in a Bayesian framework. See [11] for non-Bayesian alternatives in the context of boosting.

In terms of convex duality, parameter priors for the dual problem correspond to "potentials" on the constraint values in the primal problem. The case of a Gaussian prior on $\lambda$, for example, corresponds to a quadratic potential on the constraint values in the primal problem.

We now consider primal problems over $(p, c)$ where $p \in \mathcal{M}$ and $c \in \mathbb{R}^m$ is a parameter vector that relaxes the original constraints. Define $\mathcal{F}(\widetilde{p}, f, c) \subset \mathcal{M}$ as

$$\mathcal{F}(\widetilde{p}, f, c) \;=\; \left\{ p \in \mathcal{M} \;:\; \sum_x \widetilde{p}(x) \sum_y p(y \mid x)\, (f_j(x, y) - E_{\widetilde{p}}[f_j \mid x]) = c_j \right\} \quad (4)$$

and consider the primal problem $P_{1,\mathrm{reg}}$ given by

$$(P_{1,\mathrm{reg}}) \qquad minimize \quad D(p, q_0) + U(c)$$
$$subject\ to \quad p \in \mathcal{F}(\widetilde{p}, f, c)$$

where $U : \mathbb{R}^m \to \mathbb{R}$ is a convex function whose minimum is at 0.

To derive the dual problem, the Lagrangian is calculated as $\mathcal{L}(p, c, \lambda) = \mathcal{L}(p, \lambda) + U(c)$ and the dual function is $h_{1,\mathrm{reg}}(\lambda) = h_1(\lambda) + U^*(\lambda)$ where $U^*(\lambda)$ is the convex conjugate of $U$. For a quadratic penalty $U(c) = \sum_j \frac{1}{2} \sigma_j^2 c_j^2$, we have $U^*(\lambda) = -\sum_j \frac{1}{2} \sigma_j^{-2} \lambda_j^2$ and the dual function becomes

$$h_{1,\mathrm{reg}}(\lambda) \;=\; -\sum_x \widetilde{p}(x) \sum_y q_0(y \mid x)\, e^{\langle \lambda, f(x,y) - f(x,\tilde{y}(x)) \rangle} - \sum_j \frac{\lambda_j^2}{2\sigma_j^2}. \quad (5)$$

A sequential update rule for (5) incurs the small additional cost of solving a nonlinear equation by Newton-Raphson every iteration. See [1] for a discussion of this technique in the context of exponential models in statistical language modeling.

## 5   Experiments

We performed experiments on some of the UC Irvine datasets in order to investigate the relationship between boosting and maximum likelihood empirically. The weak learner was the decision stump `FindAttrTest` as described in [6], and the training set consisted of a randomly chosen 90% of the data. Table 1 shows experiments with regularized boosting. Two boosting models are compared. The first model $q_1$ was trained for 10 features generated by `FindAttrTest`, excluding features satisfying condition (3). Training was carried out using the parallel update method described in [2]. The second model, $q_2$, was trained using the exponential loss with quadratic regularization. The performance was measured using the conditional log-likelihood of the (normalized) models over the training and test set, denoted $\ell_{train}$ and $\ell_{test}$, as well as using the test error rate $\epsilon_{test}$. The table entries were averaged by 10-fold cross validation.

For the weak learner `FindAttrTest`, only the Iris dataset produced features that satisfy (3). On average, 4 out of the 10 features were removed. As the flexibility of the weak learner is increased, (3) is expected to hold more often. On this dataset regularization improves both the test set log-likelihood and error rate considerably. In datasets where $q_1$ shows significant overfitting, regularization improves both the log-likelihood measure and the error rate. In cases of little overfitting (according to the log-likelihood measure), regularization only improves the test set log-likelihood at the expense of the training set log-likelihood, however without affecting test set error.

Next we performed a set of experiments to test how much $q_{boost}^\star$ differs from $q_{ml}^\star$, where the boosting model is normalized (after training) to form a conditional probability distribution. For different experiments, `FindAttrTest` generated a different number of features (10–100), and the training set was selected randomly. The top row in Figure 2 shows for the Sonar dataset the relationship between $\ell_{train}(q_{ml}^\star)$ and $\ell_{train}(q_{boost}^\star)$ as well as between $\ell_{train}(q_{ml}^\star)$ and $D_{train}(q_{ml}^\star, q_{boost}^\star)$. As the number of features increases so that the training

|  | Unregularized | | | Regularized | | |
|---|---|---|---|---|---|---|
| Data | $\ell_{train}(q_1)$ | $\ell_{test}(q_1)$ | $\epsilon_{test}(q_1)$ | $\ell_{train}(q_2)$ | $\ell_{test}(q_2)$ | $\epsilon_{test}(q_2)$ |
| Promoters | -0.29 | -0.60 | 0.28 | -0.32 | -0.50 | 0.26 |
| Iris | -0.29 | -1.16 | 0.21 | -0.10 | -0.20 | 0.09 |
| Sonar | -0.22 | -0.58 | 0.25 | -0.26 | -0.48 | 0.19 |
| Glass | -0.82 | -0.90 | 0.36 | -0.84 | -0.90 | 0.36 |
| Ionosphere | -0.18 | -0.36 | 0.13 | -0.21 | -0.28 | 0.10 |
| Hepatitis | -0.28 | -0.42 | 0.19 | -0.28 | -0.39 | 0.19 |
| Breast | -0.12 | -0.14 | 0.04 | -0.12 | -0.14 | 0.04 |
| Pima | -0.48 | -0.53 | 0.26 | -0.48 | -0.52 | 0.25 |

Table 1: Comparison of unregularized to regularized boosting. For both the regularized and unregularized cases, the first column shows training log-likelihood, the second column shows test log-likelihood, and the third column shows test error rate. Regularization reduces error rate in some cases while it consistently improves the test set log-likelihood measure on all datasets. All entries were averaged using 10-fold cross validation.

data is more closely fit ($\ell_{train}(q_{ml}) \longrightarrow 0$), the boosting and maximum likelihood models become more similar, as measured by the KL divergence. This result does not hold when the model is unidentifiable and the two models diverge in arbitrary directions.

The bottom row in Figure 2 shows the relationship between the test set log-likelihoods, $\ell_{test}(q_{ml}^\star)$ and $\ell_{test}(q_{boost}^\star)$, together with the test set error rates $\epsilon_{test}(q_{ml}^\star)$ and $\epsilon_{test}(q_{boost}^\star)$. In these figures the testing set was chosen to be 50% of the total data. In order to indicate the number of points at each error rate, each circle was shifted by a small random value to avoid points falling on top of each other. While the plots in the bottom row of Figure 2 indicate that $\ell_{train}(q_{ml}^\star) > \ell_{train}(q_{boost}^\star)$, as expected, on the test data the linear trend is reversed, so that $\ell_{test}(q_{ml}^\star) < \ell_{test}(q_{boost}^\star)$. Identical experiments on Hepatitis, Glass and Promoters resulted in similar results and are omitted due to lack of space.

The duality result suggests a possible explanation for the higher performance of boosting with respect to $\ell_{test}$. The boosting model is less constrained due to the lack of normalization constraints, and therefore has a smaller $I$-divergence to the uniform model. This may be interpreted as a higher extended entropy, or less concentrated conditional model.

However, as $\ell(q_{ml}^\star) \longrightarrow 0$, the two models come to agree (up to identifiability). It is easy to show that for any exponential model $q_\lambda \in \mathcal{Q}_2$, $D_{train}(q_{ml}^\star, q_\lambda) = \ell(q_{ml}^\star) - \ell(q_\lambda)$. By taking $q_\lambda = q_{boost}^\star$ it is seen that as the difference between $\ell(q_{ml}^\star)$ and $\ell(q_{boost}^\star)$ gets smaller, the divergence between the two models also gets smaller. The empirical results are consistent with the theoretical analysis. As the number of features is increased so that the training data is fit more closely, the model matches the empirical distribution $\widetilde{p}$ and the normalizing term $Z_\lambda(x)$ becomes a constant. In this case, normalizing the boosting model $q_{boost}^\star$ does not violate the constraints, and results in the maximum likelihood model.

## Acknowledgments

We thank Michael Collins, Michael Jordan, Andrew Ng, Fernando Pereira, Rob Schapire, and Yair Weiss for helpful comments on an early version of this paper. Part of this work was carried out while the second author was visiting the Department of Statistics, University of California at Berkeley.

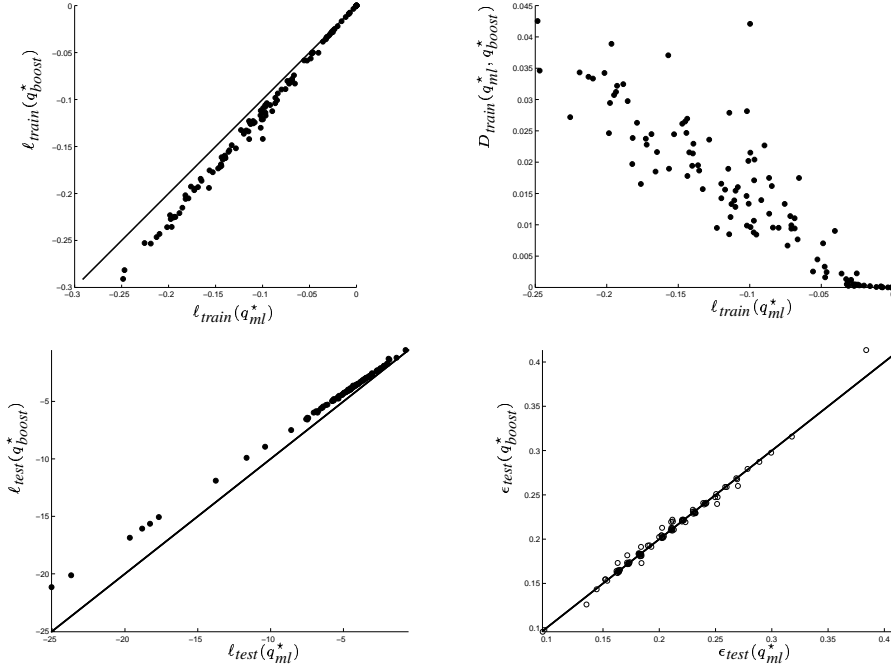

Figure 2: Comparison of AdaBoost and maximum likelihood for Sonar dataset. The top row compares $\ell_{train}(q_{ml}^{\star})$ to $\ell_{train}(q_{boost}^{\star})$ (left) and $\ell_{train}(q_{ml}^{\star})$ to $D_{train}(q_{ml}^{\star}, q_{boost}^{\star})$ (right). The bottom row shows the relationship between $\ell_{test}(q_{ml}^{\star})$ and $\ell_{test}(q_{boost}^{\star})$ (left) and $\epsilon_{test}(q_{ml}^{\star})$ and $\epsilon_{test}(q_{boost}^{\star})$ (right). The experimental results for other UCI datasets were very similar.

# References

[1] S. Chen and R. Rosenfeld. A survey of smoothing techniques for ME models. *IEEE Transactions on Speech and Audio Processing*, 8(1), 2000.

[2] M. Collins, R. E. Schapire, and Y. Singer. Logistic regression, AdaBoost and Bregman distances. *Machine Learning*, to appear.

[3] S. Della Pietra, V. Della Pietra, and J. Lafferty. Inducing features of random fields. *IEEE Transactions on Pattern Analysis and Machine Intelligence*, 19(4), 1997.

[4] S. Della Pietra, V. Della Pietra, and J. Lafferty. Duality and auxiliary functions for Bregman distances. Technical Report CMU-CS-01-109, Carnegie Mellon University, 2001.

[5] N. Duffy and D. Helmbold. Potential boosters? In *Neural Information Processing Systems*, 2000.

[6] Y. Freund and R. E. Schapire. Experiments with a new boosting algorithm. In *International Conference on Machine Learning*, 1996.

[7] J. Friedman, T. Hastie, and R. Tibshirani. Additive logistic regression: A statistical view of boosting. *The Annals of Statistics*, 28(2), 2000.

[8] J. Kivinen and M. K. Warmuth. Boosting as entropy projection. In *Computational Learning Theory*, 1999.

[9] J. Lafferty. Additive models, boosting, and inference for generalized divergences. In *Computational Learning Theory*, 1999.

[10] L. Mason, J. Baxter, P. Bartlett, and M. Frean. Functional gradient techniques for combining hypotheses. In A. Smola, P. Bartlett, B. Schölkopf, and D. Schuurmans, editors, *Advances in Large Margin Classifiers*, 1999.

[11] G. Rätsch, T. Onoda, and K.-R. Müller. Soft margins for AdaBoost. *Machine Learning*, 2001.

